# Relative Density Nets: A New Way to Combine Backpropagation with HMM's

**Andrew D. Brown**
Department of Computer Science
University of Toronto
Toronto, Canada M5S 3G4
*andy@cs.utoronto.ca*

**Geoffrey E. Hinton**
Gatsby Unit, UCL
London, UK WC1N 3AR
*hinton@gatsby.ucl.ac.uk*

## Abstract

Logistic units in the first hidden layer of a feedforward neural network compute the relative probability of a data point under two Gaussians. This leads us to consider substituting other density models. We present an architecture for performing discriminative learning of Hidden Markov Models using a network of many small HMM's. Experiments on speech data show it to be superior to the standard method of discriminatively training HMM's.

## 1 Introduction

A standard way of performing classification using a generative model is to divide the training cases into their respective classes and then train a set of class conditional models. This unsupervised approach to classification is appealing for two reasons. It is possible to reduce overfitting, because the model learns the class-conditional input densities $P(\mathbf{x}|c)$ rather than the input-conditional class probabilities $P(c|\mathbf{x})$. Also, provided that the model density is a good match to the underlying data density then the decision provided by a probabilistic model is Bayes optimal. The problem with this unsupervised approach to using probabilistic models for classification is that, for reasons of computational efficiency and analytical convenience, very simple generative models are typically used and the optimality of the procedure no longer holds. For this reason it is usually advantageous to train a classifier discriminatively.

In this paper we will look specifically at the problem of learning HMM's for classifying speech sequences. It is an application area where the assumption that the HMM is the correct generative model for the data is inaccurate and discriminative methods of training have been successful. The first section will give an overview of current methods of discriminatively training HMM classifiers. We will then introduce a new type of multi-layer backpropagation network which takes better advantage of the HMM's for discrimination. Finally, we present some simulations comparing the two methods.

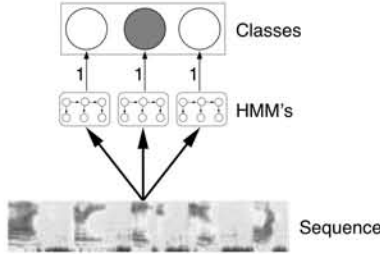

Figure 1: An Alphanet with one HMM per class. Each computes a score for the sequence and this feeds into a softmax output layer.

## 2   Alphanets and Discriminative Learning

The unsupervised way of using an HMM for classifying a collection of sequences is to use the Baum-Welch algorithm [1] to fit one HMM per class. Then new sequences are classified by computing the probability of a sequence under each model and assigning it to the one with the highest probability. Speech recognition is one of the commonest applications of HMM's, but unfortunately an HMM is a poor model of the speech production process. For this reason speech researchers have looked at the possibility of improving the performance of an HMM classifier by using information from negative examples — examples drawn from classes other than the one which the HMM was meant to model. One way of doing this is to compute the mutual information between the class label and the data under the HMM density, and maximize that objective function [2].

It was later shown that this procedure could be viewed as a type of neural network (see Figure 1) in which the inputs to the network are the log-probability scores $\mathcal{L}(\mathbf{x}_{1:T}|\mathcal{H})$ of the sequence under hidden Markov model $\mathcal{H}$ [3]. In such a model there is one HMM per class, and the output is a softmax non-linearity:

$$P(c_k|\mathbf{x}_{1:T}, \mathcal{H}_1, \ldots, \mathcal{H}_K) = y_k = \frac{\exp(\mathcal{L}(\mathbf{x}_{1:T}|\mathcal{H}_k))}{\sum_{j=1}^{K} \exp(\mathcal{L}(\mathbf{x}_{1:T}|\mathcal{H}_j))} \tag{1}$$

Training this model by maximizing the log probability of correct classification leads to a classifier which will perform better than an equivalent HMM model trained solely in a unsupervised manner. Such an architecture has been termed an "Alphanet" because it may be implemented as a recurrent neural network which mimics the forward pass of the forward-backward algorithm.[1]

## 3   Backpropagation Networks as Density Comparators

A multi-layer feedforward network is usually thought of as a flexible non-linear regression model, but if it uses the logistic function non-linearity in the hidden layer, there is an interesting interpretation of the operation performed by each hidden unit. Given a mixture of two Gaussians where we know the component priors $P(\mathcal{G})$ and the component densities $P(\mathbf{x}|\mathcal{G})$ then the posterior probability that Gaussian, $\mathcal{G}_0$, generated an observation $\mathbf{x}$, is a logistic function whose argument is the negative log-odds of the two classes [4]. This can clearly be seen by rearranging

the expression for the posterior:

$$
\begin{aligned}
P(\mathcal{G}_0|\mathbf{x}) &= \frac{P(\mathbf{x}|\mathcal{G}_0)P(\mathcal{G}_0)}{P(\mathbf{x}|\mathcal{G}_0)P(\mathcal{G}_0) + P(\mathbf{x}|\mathcal{G}_1)P(\mathcal{G}_1)} \\
&= \frac{1}{1 + \exp\left\{-\log\frac{P(\mathbf{x}|\mathcal{G}_0)}{P(\mathbf{x}|\mathcal{G}_1)} - \log\frac{P(\mathcal{G}_0)}{P(\mathcal{G}_1)}\right\}}
\end{aligned}
\tag{2}
$$

If the class conditional densities in question are multivariate Gaussians

$$
P(\mathbf{x}|\mathcal{G}_k) = |2\pi\Sigma|^{-\frac{1}{2}} \exp\left\{-\frac{1}{2}(\mathbf{x} - \mu_k)^T \Sigma^{-1}(\mathbf{x} - \mu_k)\right\}
\tag{3}
$$

with equal covariance matrices, $\Sigma$, then the posterior class probability may be written in this familiar form:

$$
P(\mathcal{G}_0|\mathbf{x}) = \frac{1}{1 + \exp\{-(\mathbf{x}^T\mathbf{w} + \mathbf{b})\}}
\tag{4}
$$

where,

$$
\mathbf{w} = \Sigma^{-1}(\mu_0 - \mu_1)
\tag{5}
$$

$$
\mathbf{b} = (\mu_1 + \mu_0)^T \Sigma(\mu_1 - \mu_0) + \log\frac{P(\mathcal{G}_0)}{P(\mathcal{G}_1)}
\tag{6}
$$

Thus, the multi-layer perceptron can be viewed as computing pairwise posteriors between Gaussians in the input space, and then combining these in the output layer to compute a decision.

## 4 A New Kind of Discriminative Net

This view of a feedforward network suggests variations in which other kinds of density models are used in place of Gaussians in the input space. In particular, instead of performing pairwise comparisons between Gaussians, the units in the first hidden layer can perform pairwise comparisons between the densities of an input sequence under $M$ different HMM's. For a given sequence the log-probability of a sequence under each HMM is computed and the difference in log-probability is used as input to the logistic hidden unit.[2] This is equivalent to computing the posterior responsibilities of a mixture of two HMM's with equal prior probabilities. In order to maximally leverage the information captured by the HMM's we use $\binom{M}{2}$ hidden units so that all possible pairs are included. The output of a hidden unit $h$ is given by

$$
h_{(mn)} = \sigma(\mathcal{L}(\mathbf{x}_{1:T}|\mathcal{H}_m) - \mathcal{L}(\mathbf{x}_{1:T}|\mathcal{H}_n))
\tag{7}
$$

where we have used $(mn)$ as an index over the set, $\binom{M}{2}$, of all unordered pairs of the HMM's. The results of this hidden layer computation are then combined using a fully connected layer of free weights, $W$, and finally passed through a softmax function to make the final decision.

$$
a_k = \sum_{(mn)\in\binom{M}{2}} w_{(m,n)k} h_{(mn)}
\tag{8}
$$

$$
P(c_k|x_{1:T}, \mathcal{H}_1, \ldots, \mathcal{H}_M) = p_k = \frac{\exp(a_k)}{\sum_{k'=1}^{K} \exp(a_{k'})}
\tag{9}
$$

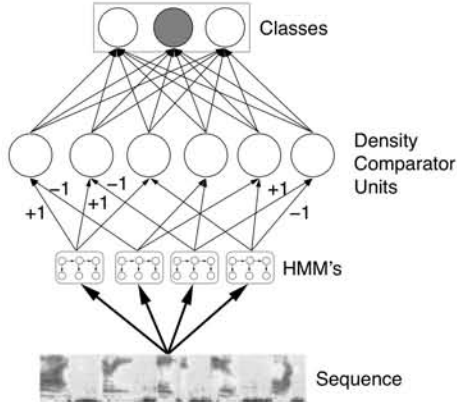

Figure 2: A multi-layer density net with HMM's in the input layer. The hidden layer units perform all pairwise comparisons between the HMM's.

where we have used $\sigma(\cdot)$ as shorthand for the logistic function, and $p_k$ is the value of the $k$th output unit. The resulting architecture is shown in figure 2. Because each unit in the hidden layer takes as input the difference in log-probability of two HMM's, this can be thought of as a fixed layer of weights connecting each hidden unit to a pair of HMM's with weights of $\pm 1$.

In contrast to the Alphanet, which allocates one HMM to model each class, this network does not require a one-to-one alignment between models and classes and it gets maximum discriminative benefit from the HMM's by comparing all pairs. Another benefit of this architecture is that it allows us to use more HMM's than there are classes. The unsupervised approach to training HMM classifiers is problematic because it depends on the assumption that a single HMM is a good model of the data and, in the case of speech, this is a poor assumption. Training the classifier discriminatively alleviated this drawback and the multi-layer classifier goes even further in this direction by allowing many HMM's to be used to learn the decision boundaries between the classes. The intuition here is that many small HMM's can be a far more efficient way to characterize sequences than one big HMM. When many small HMM's cooperate to generate sequences, the mutual information between different parts of generated sequences scales linearly with the number of HMM's and only logarithmically with the number of hidden nodes in each HMM [5].

## 5 Derivative Updates for a Relative Density Network

The learning algorithm for an RDN is just the backpropagation algorithm applied to the network architecture as defined in equations 7,8 and 9. The output layer is a distribution over class memberships of data point $\mathbf{x}_{1:T}$, and this is parameterized as a softmax function. We minimize the cross-entropy loss function:

$$\ell = \sum_{k=1}^{K} t_k \log p_k \tag{10}$$

where $p_k$ is the value of the $k$th output unit and $t_k$ is an indicator variable which is equal to 1 if $k$ is the true class. Taking derivatives of this expression with respect to the inputs of the output units yields

$$\frac{\partial \ell}{\partial a_k} = t_k - p_k \tag{11}$$

$$\frac{\partial \ell}{\partial w_{(mn),k}} = \frac{\partial \ell}{\partial a_k} \frac{\partial a_k}{\partial w_{(mn),k}} = (t_k - p_k)h_{(mn)} \tag{12}$$

The derivative of the output of the $(mn)$th hidden unit with respect to the output of $i$th HMM, $\mathcal{L}_i$, is

$$\frac{\partial h_{(mn)}}{\partial \mathcal{L}_i} = \sigma(\mathcal{L}_m - \mathcal{L}_n)(1 - \sigma(\mathcal{L}_m - \mathcal{L}_n))(\delta_{im} - \delta_{in}) \tag{13}$$

where $(\delta_{im} - \delta_{in})$ is an indicator which equals $+1$ if $i = m$, $-1$ if $i = n$ and zero otherwise. This derivative can be chained with the the derivatives backpropagated from the output to the hidden layer.

For the final step of the backpropagation procedure we need the derivative of the log-likelihood of each HMM with respect to its parameters. In the experiments we use HMM's with a single, axis-aligned, Gaussian output density per state. We use the following notation for the parameters:

- $A$: $a_{ij}$ is the transition probability from state $i$ to state $j$
- $\Pi$: $\pi_i$ is the initial state prior
- $\mu_i$: mean vector for state $i$
- $\mathbf{v}_i$: vector of variances for state $i$
- $\mathcal{H}$: set of HMM parameters $\{A, \Pi, \mu, \mathbf{v}\}$

We also use the variable $s_t$ to represent the state of the HMM at time $t$. We make use of the property of all latent variable density models that the derivative of the log-likelihood is equal to the expected derivative of the joint log-likelihood under the posterior distribution. For an HMM this means that:

$$\frac{\partial \mathcal{L}(\mathbf{x}_{1:T}|\mathcal{H})}{\partial \mathcal{H}_i} = \sum_{s_{1:T}} P(s_{1:T}|\mathbf{x}_{1:T}, \mathcal{H}) \frac{\partial}{\partial \mathcal{H}_i} \log P(\mathbf{x}_{1:T}, s_{1:T}|\mathcal{H}) \tag{14}$$

The joint likelihood of an HMM is:

$$\langle \log P(\mathbf{x}_{1:T}, s_{1:T}|\mathcal{H}) \rangle =$$

$$= \sum_i \langle \delta_{s_1,i} \rangle \log \pi_i + \sum_{t=2}^{T} \sum_{i,j} \langle \delta_{s_t,j} \delta_{s_{t-1},i} \rangle \log a_{ij} +$$

$$\sum_{t=1}^{T} \sum_i \langle \delta_{s_t,i} \rangle \left[ -\frac{1}{2} \sum_{d=1}^{D} \log v_{i,d} - \frac{1}{2} \sum_{d=1}^{D} (x_{t,d} - \mu_{i,d})^2 / v_{i,d} \right] + const \tag{15}$$

where $\langle \cdot \rangle$ denotes expectations under the posterior distribution and $\langle \delta_{s_t,i} \rangle$ and $\langle \delta_{s_t,j} \delta_{s_{t-1},i} \rangle$ are the expected state occupancies and transitions under this distribution. All the necessary expectations are computed by the forward backward algorithm. We could take derivatives with respect to this functional directly, but that would require doing constrained gradient descent on the probabilities and the variances. Instead, we reparameterize the model using a softmax basis for probability vectors and an exponential basis for the variance parameters. This choice of basis allows us to do unconstrained optimization in the new basis. The new parameters are defined as follows:

$$a_{ij} = \frac{\exp(\theta_{ij}^{(a)})}{\sum_{j'} \exp(\theta_{ij'}^{(a)})}, \quad \pi_i = \frac{\exp(\theta_i^{(\pi)})}{\sum_{i'} \exp(\theta_i^{(\pi)})}, \quad v_{i,d} = \exp(\theta_{i,d}^{(\mathbf{v})})$$

This results in the following derivatives:

$$\frac{\partial \mathcal{L}(\mathbf{x}_{1:T}|\mathcal{H})}{\partial \theta_{ij}^{(a)}} = \sum_{t=2}^{T} \left[ \langle \delta_{s_t,j} \delta_{s_{t-1},i} \rangle - \langle \delta_{s_{t-1},i} \rangle a_{ij} \right] \tag{16}$$

$$\frac{\partial \mathcal{L}(\mathbf{x}_{1:T}|\mathcal{H})}{\partial \theta_i^{(\pi)}} = \langle \delta_{s_1,i} \rangle - \pi_i \tag{17}$$

$$\frac{\partial \mathcal{L}(\mathbf{x}_{1:T}|\mathcal{H})}{\partial \mu_{i,d}} = \sum_{t=1}^{T} \langle \delta_{s_t,i} \rangle (x_{t,d} - \mu_{i,d})/v_{i,d} \tag{18}$$

$$\frac{\partial \mathcal{L}(\mathbf{x}_{1:T}|\mathcal{H})}{\partial \theta_{i,d}^{(\mathbf{v})}} = \frac{1}{2} \sum_{t=1}^{T} \langle \delta_{s_t,i} \rangle \left[ (x_{t,d} - \mu_{i,d})^2/v_{i,d} - 1 \right] \tag{19}$$

When chained with the error signal backpropagated from the output, these derivatives give us the direction in which to move the parameters of each HMM in order to increase the log probability of the correct classification of the sequence.

## 6  Experiments

To evaluate the relative merits of the RDN, we compared it against an Alphanet on a speaker identification task. The data was taken from the CSLU 'Speaker Recognition' corpus. It consisted of 12 speakers uttering phrases consisting of 6 different sequences of connected digits recorded multiple times (48) over the course of 12 recording sessions. The data was pre-emphasized and Fourier transformed in 32ms frames at a frame rate of 10ms. It was then filtered using 24 bandpass, mel-frequency scaled filters. The log magnitude filter response was then used as the feature vector for the HMM's. This pre-processing reduced the data dimensionality while retaining its spectral structure.

While mel-cepstral coefficients are typically recommended for use with axis-aligned Gaussians, they destroy the spectral structure of the data, and we would like to allow for the possibility that of the many HMM's some of them will specialize on particular sub-bands of the frequency domain. They can do this by treating the variance as a measure of the importance of a particular frequency band — using large variances for unimportant bands, and small ones for bands to which they pay particular attention.

We compared the RDN with an Alphanet and three other models which were implemented as controls. The first of these was a network with a similar architecture to the RDN (as shown in figure 2), except that instead of fixed connections of $\pm 1$, the hidden units have a set of adaptable weights to all $M$ of the HMM's. We refer to this network as a comparative density net (CDN). A second control experiment used an architecture similar to a CDN without the hidden layer, $i.e.$ there is a single layer of adaptable weights directly connecting the HMM's with the softmax output units. We label this architecture a CDN-1. The CDN-1 differs from the Alphanet in that each softmax output unit has adaptable connections to the HMM's and we can vary the number of HMM's, whereas the Alphanet has just one HMM per class directly connected to each softmax output unit. Finally, we implemented a version of a network similar to an Alphanet, but using a mixture of Gaussians as the input density model. The point of this comparison was to see if the HMM actually achieves a benefit from modelling the temporal aspects of the speaker recognition task.

In each experiment an RDN constructed out of a set of, $M$, 4-state HMM's was compared to the four other networks all matched to have the same number of free parameters, except for the MoGnet. In the case of the MoGnet, we used the same number of Gaussian mixture models as HMM's in the Alphanet, each with the same number of hidden states. Thus, it has fewer parameters, because it is lacking the transition probabilities of the HMM. We ran the experiment four times with

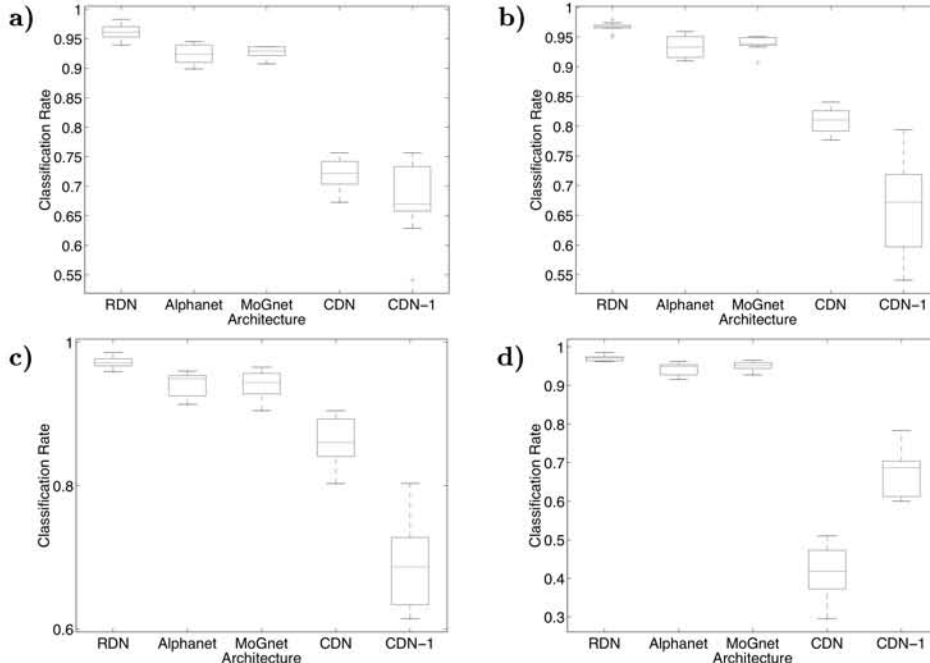

Figure 3: Results of the experiments for an RDN with (a) 12, (b) 16, (c) 20 and (d) 24 HMM's.

values of $M$ of 12, 16, 20 and 24. For the Alphanet and MoGnet we varied the number of states in the HMM's and the Gaussian mixtures, respectively. For the CDN model we used the same number of 4-state HMM's as the RDN and varied the number of units in the hidden layer of the network. Since the CDN-1 network has no hidden units, we used the same number of HMM's as the RDN and varied the number of states in the HMM. The experiments were repeated 10 times with different training-test set splits. All the models were trained using 90 iterations of a conjugate gradient optimization procedure [6].

## 7   Results

The boxplot in figure 3 shows the results of the classification performance on the 10 runs in each of the 4 experiments. Comparing the Alphanet and the RDN we see that the RDN consistently outperforms the Alphanet. In all four experiments the difference in their performance under a paired t-test was significant at the level $p < 0.01$. This indicates that given a classification network with a fixed number of parameters, there is an advantage to using many small HMM's and using all the pairwise information about an observed sequence, as opposed to using a network with a single large HMM per class.

In the third experiment involving the MoGnet we see that its performance is comparable to that of the Alphanet. This suggests that the HMM's ability to model the temporal structure of the data is not really necessary for the speaker classification task as we have set it up.[3] Nevertheless, the performance of both the Alphanet and

the MoGnet is less than the RDN.

Unfortunately the CDN and CDN-1 networks perform much worse than we expected. While we expected these models to perform similarly to the RDN, it seems that the optimization procedure takes much longer with these models. This is probably because the small initial weights from the HMM's to the next layer severely attenuate the backpropagated error derivatives that are used to train the HMM's. As a result the CDN networks do not converge properly in the time allowed.

## 8  Conclusions

We have introduced relative density networks, and shown that this method of discriminatively learning many small density models in place of a single density model per class has benefits in classification performance. In addition, there may be a small speed benefit to using many smaller HMM's compared to a few big ones. Computing the probability of a sequence under an HMM is order $O(TK^2)$, where $T$ is the length of the sequence and $K$ is the number of hidden states in the network. Thus, smaller HMM's can be evaluated faster. However, this is somewhat counterbalanced by the quadratic growth in the size of the hidden layer as $M$ increases.

### Acknowledgments

We would like to thank John Bridle, Chris Williams, Radford Neal, Sam Roweis, Zoubin Ghahramani, and the anonymous reviewers for helpful comments.

## Footnotes

[1] The results of the forward pass are the probabilities of the hidden states conditioned on the past observations, or "alphas" in standard HMM terminology.

[2]We take the time averaged log-probability so that the scale of the inputs is independent of the length of the sequence.

[3]If we had done text-dependent speaker identification, instead of multiple digit phrases

## References

[1] L. E. Baum, T. Petrie, G. Soules, and N. Weiss, "A maximization technique occurring in the statistical analysis of probabilistic functions of Markov chains," *The Annals of Mathematical Statistics*, vol. 41, no. 1, pp. 164–171, 1970.

[2] L. R. Bahl, P. F. Brown, P. V. de Souza, and R. L. Mercer, "Maximum mutual information of hidden Markov model parameters for speech recognition," in *Proceeding of the IEEE International Conference on Acoustics, Speech and Signal Processing*, pp. 49–53, 1986.

[3] J. Bridle, "Training stochastic model recognition algorithms as networks can lead to maximum mutual information estimation of parameters," in *Advances in Neural Information Processing Systems* (D. Touretzky, ed.), vol. 2, (San Mateo, CA), pp. 211–217, Morgan Kaufmann, 1990.

[4] M. I. Jordan, "Why the logistic function? A tutorial discussion on probabilities and neural networks," Tech. Rep. Computational Cognitive Science, Technical Report 9503, Massachusetts Institute of Technology, August 1995.

[5] A. D. Brown and G. E. Hinton, "Products of hidden Markov models," in *Proceedings of Artificial Intelligence and Statistics 2001* (T. Jaakkola and T. Richardson, eds.), pp. 3–11, Morgan Kaufmann, 2001.

[6] C. E. Rasmussen, *Evaluation of Gaussian Processes and other Methods for Non-Linear Regression*. PhD thesis, University of Toronto, 1996. Matlab conjugate gradient code available from http://www.gatsby.ucl.ac.uk/~edward/code/.

---

then this might have made a difference.